# Learning to Take Concurrent Actions

**Khashayar Rohanimanesh**
Department of Computer Science
University of Massachusetts
Amherst, MA 01003
*khash@cs.umass.edu*

**Sridhar Mahadevan**
Department of Computer Science
University of Massachusetts
Amherst, MA 01003
*mahadeva@cs.umass.edu*

## Abstract

We investigate a general semi-Markov Decision Process (SMDP) framework for modeling concurrent decision making, where agents learn optimal plans over concurrent temporally extended actions. We introduce three types of parallel termination schemes – *all*, *any* and *continue* – and theoretically and experimentally compare them.

## 1 Introduction

We investigate a general framework for modeling *concurrent* actions. The notion of concurrent action is formalized in a general way, to capture both situations where a single agent can execute multiple parallel processes, as well as the multi-agent case where many agents act in parallel. Concurrency clearly allows agents to achieve goals more quickly: in making breakfast, we interleave making toast and coffee with other activities such as getting milk; in driving, we search for road signs while controlling the wheel, accelerator and brakes.

Most previous work on concurrency has focused on parallelizing primitive (unit step) actions. Reiter developed axioms for concurrent planning using the situation calculus framework [4]. Knoblock [3] and Boutilier [1] modify the STRIPS representation of actions to allow for concurrent actions. These approaches assume deterministic effects. Prior work in decision-theoretic planning includes work on multi-dimensional vector action spaces [2], and models based on dynamic merging of multiple MDPs [6]. There is also a massive literature on concurrent processes, dynamic logic, and temporal logic. Parts of these lines of research deal with the specification and synthesis of concurrent actions, including probabilistic ones [8].

In contrast, we focus on parallelizing temporally extended actions. The concurrency framework described below significantly extends our previous work [5]. We provide a detailed analysis of three termination schemes for composing parallel action structures. The three schemes – *any*, *all*, and *continue* – are illustrated in Figure 1. We characterize the class of policies under each scheme. We also theoretically compare the optimality of the concurrent policies under each scheme with that of the typical

sequential case. The theoretical results are complemented by an experimental study, which illustrate the trade-offs between optimality and convergence speed, and the advantages of concurrency over sequentiality.

## 2    Concurrent Action Model

Building on SMDPs, we introduce the *Concurrent Action Model (CAM)* $(\mathcal{S}, \mathcal{A}, \mathcal{T}, \mathcal{R})$, where $\mathcal{S}$ is a set of states, $\mathcal{A}$ is a set of *primary* actions, $\mathcal{T}$ is a transition probability distribution $\mathcal{S} \times \wp(\mathcal{A}) \times \mathcal{S} \times \mathbf{N} \to [0, 1]$, where $\wp(\mathcal{A})$ is the power-set of the primary actions and $\mathbf{N}$ is the set of natural numbers, and $\mathcal{R}$ is the reward function mapping $\mathcal{S} \to \Re$. Here, a concurrent action is simply represented as a set of primary actions (hereafter called a *multi-action*), where each primary action is either a single step action, or a *temporally extended action* (e.g., modeled as a closed loop policy over single step actions [7]).

We denote the set of multi-actions that can be executed in a state $s$ by $A(s)$. In practice, this function can capture *resource* constraints that limit how many actions an agent can execute in parallel. Thus, the transition probability distribution in practice may be defined over a much smaller subset than the power-set of primary actions (e.g., in the grid world example in Figure 3, the power set is $> 100$, but the set of concurrent actions is only $\approx 10$).

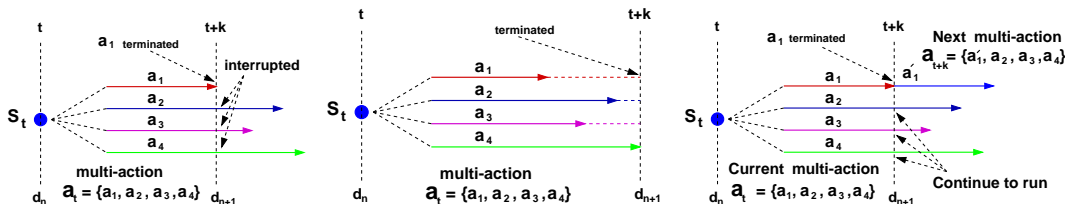

Figure 1: Left: $T_{any}$ termination scheme. Middle: $T_{all}$ termination scheme. Right: $T_{continue}$ termination scheme.

A principal goal of this paper is to understand how to define decision epochs for concurrent processes, since the primary actions in a multi-action may not terminate at the same time. The event of termination of a multi-action can be defined in many ways. Three termination schemes are illustrated in Figure 1. In the $T_{any}$ termination scheme (Figure 1, left), the next decision epoch is when the first primary action within the multi-action currently being executed terminates, where the rest of the primary actions that did not terminate naturally are interrupted (the notion of interruption is similar to [7]). In the $T_{all}$ termination scheme (Figure 1, middle), the next decision epoch is the earliest time at which all the primary actions within the multi-action currently being executed have terminated.

We can design other termination schemes by combining $T_{any}$ and $T_{all}$ : for example, another termination scheme called *continue* is one that always terminates based on the $T_{any}$ termination scheme, but lets those primary actions that did not terminate naturally continue running, while initiating new primary actions if they are going to be useful (Figure 1, right).

A deterministic *Markovian* (memoryless) policy in CAMs is defined as the mapping $\pi : \mathcal{S} \to \wp(\mathcal{A})$. Note that even though the mapping is defined independent of the

termination scheme, the behavior of a multi-action policy depends on the termination scheme that is used in the model. To illustrate this, let $< \pi, \tau >$ (called a *policy-termination* construct) denote the process of executing the multi-action policy $\pi$ using the termination scheme $\tau \in \{T_{any}, T_{all}\}$. To simplify notation, we only use this form whenever we want to explicitly point out what termination scheme is being used for executing the policy $\pi$. For a given Markovian policy, we can write the value of that policy in an arbitrary state given the termination mechanism used in the model. Let $\Theta(\pi, s_t, \tau)$ denote the event of initiating the multi-action $\pi(s_t)$ at time $t$ and terminating it according to the $\tau \in \{T_{any}, T_{all}\}$ termination scheme. Also let $\pi^{*\tau}$ denote the optimal multi-action policy within the space of policies over multi-actions that terminate according to the $\tau \in \{T_{any}, T_{all}\}$ termination scheme. To simplify notation, we may alternatively use $*_\tau$ to denote optimality with respect to the $\tau$ termination scheme. Then the optimal value function can be written as:

$$V^{*\tau}(s_t) = E\{r_{t+1} + \gamma r_{t+2} + ... + \gamma^{k-1} r_{t+k} + \gamma^k \max_{\mathbf{a} \in A(s_{t+k})} Q^{*\tau}(s_{t+k}, \mathbf{a}) \mid \Theta(\pi^{*\tau}, s_t, \tau)\}$$

where $Q^{*\tau}(s_{t+k}, \mathbf{a})$ denotes the multi-action value of executing $\mathbf{a}$ in state $s_{t+k}$ (terminated using $\tau$) and following the optimal policy $\pi^{*\tau}$ thereafter.

The policy associated with the *continue* termination scheme is a *history dependent* policy, since for a given state $s_t$, the *continue* policy will select a multi-action such that it includes the set of all the primary actions of the multi-action executed in the previous decision epoch that did not terminate naturally in the current state $s_t$ (we refer to this set as the *continue-set* represented by $h_t$). The *continue* policy is defined as the mapping $\pi_{cont} : \mathcal{S} \times \mathcal{H} \to \wp(\mathcal{A})$ in which $\mathcal{H}$ is a set of continue-sets $h_t$. Note that the value function definition for the *continue* policy should be defined over both state $s_t$ and the continue-set $h_t$ (represented by $\prec s_t, h_t \succ$), i.e., $V^{\pi_{cont}}(\prec s_t, h_t \succ)$. Let the function $A(s_t, h_t)$ return the set of multi-actions that can be executed in state $s_t$ that include the continuing primary actions in $h_t$. Then the *continue* policy is formally defined as: $\pi_{cont}(\prec s_t, h_t \succ) = arg \max_{\mathbf{a} \in A(s_t, h_t)} Q^{\pi_{cont}}(\prec s_t, h_t \succ, \mathbf{a})$.

To illustrate this, assume that the current state is $s_t$ and the multi-action $\mathbf{a_t} = \{a_1, a_2, a_3, a_4\}$ is executed in state $s_t$. Also, assume that the primary action $a_1$ is the first action that terminates after $k$ steps in state $s_{t+k}$. According to the definition of the *continue* termination scheme (that terminates based on $T_{any}$), the multi-action $\mathbf{a_t}$ is terminated at time $t + k$ and we need to select a new multi-action to execute in state $s_{t+k}$ (with the continue-set $h_{t+k} = \{a_2, a_3, a_4\}$). The *continue* policy will select the best multi-action $\mathbf{a_{t+k}}$ that includes the primary actions $\{a_2, a_3, a_4\}$, since they did not terminate in state $s_{t+k}$ (see Figure 1, right).

## 3 Theoretical Results

In this section we present some of our theoretical results comparing the optimality of various policies under different termination schemes introduced in the previous section. In all of these theorems we use the partial ordering relation $V^{\pi_1} \leq V^{\pi_2} \leftrightarrow \pi_1 \leq \pi_2$, in order to compare different policies. For lack of space, we abbreviated the proofs. Note that in theorems 1 and 3 which compare the *continue* policy with $\pi^{*any}$ and $\pi^{*all}$ policies, the value function is written over the pair $\prec s_t, h_t \succ$ to be consistent with the definition of the *continue* policy. This does not influence the original definition of the value function for the optimal policies in $T_{any}$ and $T_{all}$

termination schemes, since they are independent of the continue-set $h_t$. First, we compare the optimal multi-action policies based on the $T_{any}$ termination scheme and the *continue* policy.

**Theorem 1:** For every state $s_t \in S$, and all continue-set $h_t \in \mathcal{H}$,
$V^{\pi_{cont}}(\prec s_t, h_t \succ) \leq V^{*any}(\prec s_t, h_t \succ)$.
**Proof**: By writing the value function definition for each case we have:

$$V^{\pi_{cont}}(\prec s_t, h_t \succ) = \max_{\mathbf{a} \in A(s_t, h_t)} Q^{\pi_{cont}}(\prec s_t, h_t \succ, \mathbf{a}) \leq \max_{\mathbf{a} \in A(s_t)} Q^{\pi_{cont}}(\prec s_t, h_t \succ, \mathbf{a})$$
$$\leq \max_{\mathbf{a} \in A(s_t)} Q^{*any}(\prec s_t, h_t \succ, \mathbf{a}) = V^{*any}(\prec s_t, h_t \succ)$$

The inequality holds since the maximization in $\pi_{cont}$ is over a smaller set (i.e., $A(s_t, h_t)$) which is a subset of the larger set $A(s_t)$ that is maximized over, in the $\pi^{*any}$ case.

Next, we show that the optimal plans with multi-actions that terminate according to the $T_{any}$ termination scheme are better compared to the optimal plans with multi-actions that terminate according to the $T_{all}$ termination scheme:

**Theorem 2:** For every state $s \in S$, $V^{*all}(s) \leq V^{*any}(s)$.
**Proof**: The proof is based on the following lemma which states that if we alter the execution of the optimal multi-action policy based on $T_{all}$ (i.e., $\pi^{*all}$) in such a way that at every decision epoch the next multi-action is still selected from $\pi^{*all}$, but we terminate it based on $T_{any}$ then the new policy-termination construct represented by $< *_{all}, any >$ is better than the $\pi^{*all}$ policy. Intuitively this makes sense, since if we interrupt $\pi^{*all}(s)$ when the first primary action $a_i \in \mathbf{a} = \pi^{*all}(s)$ terminates in some future state $s'$, due to the optimality of $\pi^{*all}$, executing $\pi^{*all}(s')$ is always better than or equal to continuing some other policy such as the one in progress (i.e., $\pi^{*all}(s)$). Note that the proof is not as simple as in the first theorem since the two different policies discussed in this theorem (i.e., $\pi^{*any}$ and $\pi^{*all}$) are not being executed using the same termination method.

**Lemma 1:** For every state $s \in S$, $V^{*all}(s) \leq V^{<*_{all}, any>}(s)$.
**Proof:** Let $V_{n,any}^{*all}(s)$ denote the value of following the optimal $\pi^{*all}$ policy in state $s$, where for the first $n$ decision epochs we use the $T_{any}$ termination scheme and for the rest we use the $T_{all}$ termination scheme. By induction on $n$, we can show that $V^{*all}(s) \leq V_{n,any}^{*all}(s), \forall s \in S$ and for all $n$. This suggests that if we always terminate a multi-action $\pi^{*all}(s_t)$ according to the $T_{any}$ termination scheme, we achieve a better return; or mathematically $V^{*all}(s) \leq \lim_{n \to \infty} V_{n,any}^{*all}(s) = V^{<*_{all}, any>}(s)$.

Using Lemma 1, and the optimality of $\pi^{*any}$ in the space of policies with termination scheme according to $T_{any}$, it follows that $V^{*all}(s) \leq V^{<*_{all}, any>}(s) \leq V^{*any}(s)$.

Next, we show that if we execute the *continue* policy in which at any decision epoch we always execute the best set of primary actions along with those ones that were executed in the previous decision epoch and have not terminated yet, we achieve a better return compared to the case in which we execute the best set of primary actions, but always wait until all of the primary actions terminate before making a new decision:

**Theorem 3:** For every state $s_t \in S$, and all continue-set $h_t \in \mathcal{H}$,
$V^{*all}(\prec s_t, h_t \succ) \leq V^{\pi_{cont}}(\prec s_t, h_t \succ)$.
**Proof:** In $\pi^{*all}$ policies, multi-actions are executed until all of the primary actions

of that multi-action terminate. The *continue* policy, however, may also initiate new useful primary action in addition to those already running which may achieve a better return. Let $V_{n,cont}^{*all}(\prec s_t, h_t \succ)$ denote the value of the altered policy $\pi^{*all}$ that works as follows: for a given state and continue-set $\prec s_t, h_t \succ$, the policy $\pi^{*all}(\prec s_t, h_t \succ)$ is executed while for the first $n$ decision epochs we use the *continue* termination scheme (which means terminating according to $T_{any}$, and selecting the next multi-action according to the *continue* policy) and for the rest we use the $T_{all}$ termination scheme. By induction on $n$, it can be shown that $V^{*all}(\prec s_t, h_t \succ) \leq V_{n,cont}^{*all}(\prec s_t, h_t \succ)$ for all $n$. This suggests that as we increase $n$, the altered policy behaves more like the *continue* policy and thus in the limit we have $V^{*all}(\prec s_t, h_t \succ) \leq \lim_{n \to \infty} V_{n,cont}^{*all}(\prec s_t, h_t \succ) = V^{\pi_{cont}}(\prec s_t, h_t \succ)$ which proves the theorem.

Finally we show that the optimal multi-action policies based on $T_{all}$ termination scheme are as good as the case where the agent always executes a single primary action at a time, as it is the case in standard SMDPs. Note that this theorem does not state that concurrent plans are always better than sequential ones; it simply says that if in a problem, the sequential execution of the primary actions is the best policy, CAM is able to represent and find that policy. Let $\pi^{*seq}$ represent the optimal policy in the sequential case, where only one primary action can be executed at a time:

**Theorem 4:** For every state $s \in S$, $V^{*seq}(s) \leq V^{*all}(s)$, in which $V^{*seq}(s)$ is the value of the optimal policy when the primary actions are executed one at a time sequentially.

**Proof:** It suffices to show that sequential policies are within the space of concurrent policies. This holds since a single primary action can be considered as a multi-action containing only one primary action whose termination is consistent with either of the multi-action termination schemes (i.e., in the sequential case both $T_{any}$ and $T_{all}$ termination schemes are same).

Corollary 1 summarizes our theoretical results. It shows how different policies in a concurrent action model using different termination schemes compare to each other in terms of optimality.

**Corollary 1:** In a concurrent action model and a set of termination schemes $\{T_{any}, T_{all}, continue\}$, the following partial ordering holds among the optimal policy based on $T_{any}$, the optimal policy based on $T_{all}$, the *continue* policy and the optimal sequential policy: $\pi^{*seq} \leq \pi^{*all} \leq \pi_{cont} \leq \pi^{*any}$.

**Proof:** This follows immediately from the above theorems.

Figure 2 visually describes the summary of results that we presented in Corollary 1. According to this figure, the optimal multi-action policies based on $T_{any}$ and $T_{all}$, and also *continue* multi-action policies dominate (with respect to the partial ordering relation defined over policies) the optimal policies over the sequential case. Furthermore, policies based on *continue* multi-actions dominate the optimal multi-action policies based on $T_{all}$ termination scheme, while themselves being dominated by the optimal multi-action policies based on $T_{any}$ termination scheme.

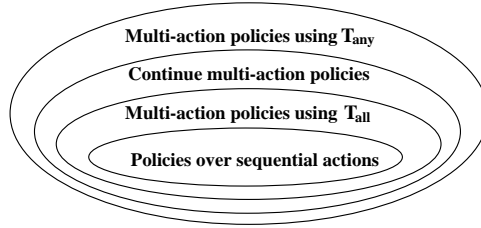

Figure 2: Comparison of policies over multi-actions and sequential primary actions using different termination schemes.

# 4   Experimental Results

In this section we present experimental results using a grid world task comparing various termination schemes (see Figure 3). Each hallway connects two rooms, and has a door with two locks. An agent has to retrieve two keys and hold both keys at the same time in order to open both locks. The process of picking up keys is modeled as a temporally extended action that takes different amount of times for each key. Moreover, keys cannot be held indefinitely, since the agent may drop a key occasionally. Therefore the agent needs to find an efficient solution for picking up the keys in parallel with navigation to act optimally. This is an episodic task, in which at the beginning of each episode the agent is placed in a fixed position (upper left corner) and the goal of the agent is to navigate to a fixed position goal (hallway H3).

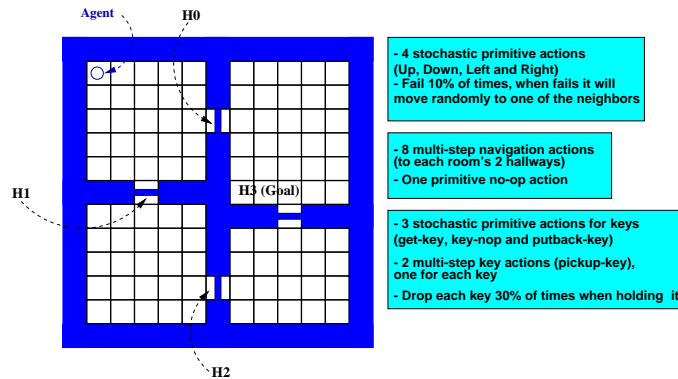

Figure 3: A navigation problem that requires concurrent plans. There are two locks on each door, which need to be opened simultaneously. Retrieving each key takes different amounts of time.

The agent can execute two types of action concurrently: (1) *navigation* actions, and (2) *key* actions. Navigation actions include a set of one-step stochastic navigation actions (Up, Left, Down and Right) that move the agent in the corresponding direction with probability 0.9 and fail with probability 0.1. Upon failure the agent moves instead in one of the other three directions, each with probability $\frac{1}{30}$. There is also a set of temporally extended actions defined over the one step navigation actions that transport the agent from within the room to one of the two hallway cells leading out of the room (Figure 4 (left)). Key actions are defined to manipulate each

key (get-key, putback-key, pickup-key, etc). Among them *pickup-key* is a temporally extended action (Figure 4 (right)). Note that each key has its own set of actions.

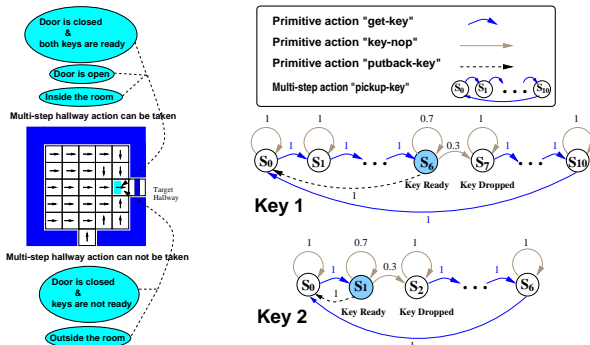

Figure 4: Left: the policy associated with one of the hallway temporally extended actions. Right: representation of the key pickup actions for each key process.

In this example, navigation actions can be executed concurrently with key actions. Actions that manipulate different keys can be also executed concurrently. However, the agent is not allowed to execute more than one navigation action, or more than one key action (from the same key action set) concurrently. In order to properly handle concurrent execution of actions, we have used a factored state space defined by state variables *position* (104 positions), *key1-state* (11 states) and *key2-state* (7 states).

In our previous work we showed that concurrent actions formed an SMDP over primitive actions [5], which turns out to hold for all the termination schemes described above. Thus, we can use SMDP Q-learning to compare concurrent policies over different termination schemes with the use of this method for purely sequential policy learning [7]. After each decision epoch where the multi-action $\mathbf{a}$ is taken in some state $s$ and terminates in state $s'$, the following update rule is used: $Q(s, \mathbf{a}) \leftarrow Q(s, \mathbf{a}) + \alpha \left[ r + \gamma^k \max_{\mathbf{a}' \in A(s')} Q(s', \mathbf{a}') - Q(s, \mathbf{a}) \right]$, where $k$ denotes the number of time steps since initiation of the multi-action $\mathbf{a}$ at state $s$ and its termination at state $s'$, and $r$ denotes the cumulative discounted reward over this period. The agent is punished by $-1$ for each primitive action. Figure 5 (left) compares the number of primitive actions taken until success, and Figure 5 (right) shows the median number of decision epochs per trial, where for trial n, it is the median of all trials from 1 to n. These data are averaged over 10 episodes, each consisting of $500,000$ trials. As shown in figure 5 (left), concurrent actions over any termination scheme yield a faster plan than sequential execution. Moreover, the policies learned based on $T_{any}$ (i.e. both $\pi^{*any}$ and $\pi_{cont}$) are also faster than $T_{all}$ . Also, $\pi^{*any}$ achieves higher optimality than $\pi_{cont}$, however the difference is small.

We conjecture that sequential execution and $T_{all}$ converge faster compared to $T_{any}$, due to the frequency with which multi-actions are terminated. As shown in Figure 5 (right), $T_{all}$ makes fewer decisions, compared to $T_{any}$. This is intuitive since $T_{all}$ terminates only when all of the primary actions in a multi-action are completed, and hence it involves less interruption compared to learning based on $T_{any}$. Note $\pi_{cont}$ converges faster than $\pi^{*any}$ and it is nearly as good as $T_{any}$. . We can think of

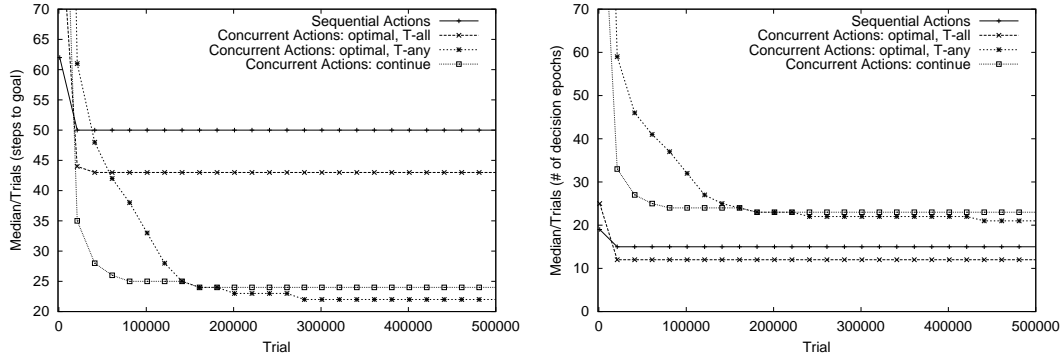

Figure 5: Left: moving median of number of steps to the goal. Right: moving median of number of multi-action level decision epochs taken to the goal.

$\pi_{cont}$ as a blend of $T_{all}$ and $T_{any}$. Even though it uses the $T_{any}$ termination scheme, it continues executing primary actions that did not terminate naturally when the first primary action terminates, making it similar to $T_{all}$.

## 5 Future Work

Even though specifying the $A(s)$ set of applicable multi-actions might significantly reduce the set of choices, we still may need additional mechanisms for efficiently searching the space of multi-actions that can run in parallel. Also, we can additionally exploit the hierarchical structure of multi-actions to compile them into an effective policy over primary actions. These are some of the practical issues that we will investigate in future work.

## References

[1] Craig Boutilier and Ronen Brafman. Planning with concurrent interacting actions. *In Proceedings of the Fourteenth National Conference on Artificial Intelligence (AAAI '97)*, 1997.

[2] P. Cichosz. Learning multidimensional control actions from delayed reinforcements. In *Eighth International Symposium on System-Modelling-Control (SMC-8)*, Zakopane, Poland, 1995.

[3] C. A. Knoblock. Generating parallel execution plans with a partial-order planner. *In Proceedings of the Second International Conference on Artificial Intelligence Planning Systems , Chicago, IL, 1994.*, 1994.

[4] Ray Reiter. Natural actions, concurrency and continuous time in the situation calculus. *Principles of Knowledge Representation and Reasoning: Proceedings of the Fifth International Conference (KR'96), Cambridge MA., November 5-8, 1996*, 1996.

[5] Khashayar Rohanimanesh and Sridhar Mahadevan. Decision-theoretic planning with concurrent temporally extended actions. *In Proceedings of the 17th Conference on Uncertainty in Artificial Intelligence*, 2001.

[6] S. Singh and David Cohn. How to dynamically merge markov decision processes. *Proceedings of NIPS 11*, 1998.

[7] R. Sutton, D. Precup, and S. Singh. Between MDPs and Semi-MDPs: A framework for temporal abstraction in reinforcement learning. *Artificial Intelligence*, pages 181–211, 1999.

[8] Glynn Winskel. Topics in concurrency: Part ii comp. sci. lecture notes. *Computer Science course at the University of Cambridge*, 2002.
